# Complex-Cell Responses Derived from Center-Surround Inputs: The Surprising Power of Intradendritic Computation

**Bartlett W. Mel and Daniel L. Ruderman**
Department of Biomedical Engineering
University of Southern California
Los Angeles, CA 90089

**Kevin A. Archie**
Neuroscience Program
University of Southern California
Los Angeles, CA 90089

## Abstract

Biophysical modeling studies have previously shown that cortical pyramidal cells driven by strong NMDA-type synaptic currents and/or containing dendritic voltage-dependent $Ca^{++}$ or $Na^+$ channels, respond more strongly when synapses are activated in several spatially clustered groups of optimal size—in comparison to the same number of synapses activated diffusely about the dendritic arbor [8]. The nonlinear intradendritic interactions giving rise to this "cluster sensitivity" property are akin to a layer of virtual nonlinear "hidden units" in the dendrites, with implications for the cellular basis of learning and memory [7, 6], and for certain classes of nonlinear sensory processing [8]. In the present study, we show that a single neuron, with access only to excitatory inputs from unoriented ON- and OFF-center cells in the LGN, exhibits the principal nonlinear response properties of a "complex" cell in primary visual cortex, namely orientation tuning coupled with translation invariance and contrast insensitivity. We conjecture that this type of intradendritic processing could explain how complex cell responses can persist in the absence of oriented simple cell input [13].

# 1  INTRODUCTION

Simple and complex cells were first described in visual cortex by Hubel and Wiesel [4]. Simple cell receptive fields could be subdivided into ON and OFF subregions, with spatial summation within a subregion and antagonism between subregions; cells of this type have historically been modeled as linear filters followed by a thresholding nonlinearity (see [13]). In contrast, complex cell receptive fields cannot generally be subdivided into distinct ON and OFF subfields, and as a group exhibit a number of fundamentally nonlinear behaviors, including (1) orientation tuning across a receptive field much wider than an optimal bar, (2) larger responses to thin bars than thick bars—in direct violation of the superposition principle, and (3) sensitivity to both light and dark bars across the receptive field.

The traditional Hubel-Wiesel model for complex cell responses involves a hierarchy, consisting of center-surround inputs that drive simple cells, which in turn provide oriented, phase-dependent input to the complex cell. By pooling over a set of simple cells with different positions and phases, the complex cell could respond selectively to stimulus orientation, while generalizing over stimulus position and contrast. A pure hierarchy involving simple cells is challenged, however, by a variety of more recent experimental results indicating many complex cells receive monosynaptic input from LGN cells [3], or do not depend on simple cell input [10, 5, 1]. It remains unknown how complex cell responses might derive from intracortical network computations that do no depend on simple cells, or whether they could originate directly from intracellular computations.

Previous biophysical modeling studies have indicated that the input-output function of a dendritic tree containing excitatory voltage-dependent membrane mechanisms can be abstracted as low-order polynomial function, i.e. a big sum of little products (see [9] for review). The close match between this type of computation and "energy" models for complex cells [12, 11, 2] suggested that a single-cell origin of complex cell responses was possible.

In the present study, we tested the hypothesis that local nonlinear processing in the dendritic tree of a single neuron, which receives only excitatory synaptic input from unoriented center-surround LGN cells, could in and of itself generate nonlinear complex cell response properties, including orientation selectivity, coupled with position and contrast invariance.

# 2  METHODS

## 2.1  BIOPHYSICAL MODELING

Simulations of a layer 5 pyramidal cell from cat visual cortex (fig. 1) were carried out in NEURON[1]. Biophysical parameters and other implementation details were as in [8] and/or shown in Table 2, except dendritic spines were not modeled here. The soma contained modified Hodgkin-Huxley channels with peak somatic conductances of $\bar{g}_{Na}$ and $\bar{g}_{DR}$ 0.20 S/cm$^2$ and 0.12 S/cm$^2$, respectively; dendritic membrane was electrically passive. Each synapse included both an NMDA and AMPA-type

[1]NEURON simulation environment courtesy Michael Hines and John Moore; synaptic channel implementations courtesy Alan Destexhe and Zach Mainen.

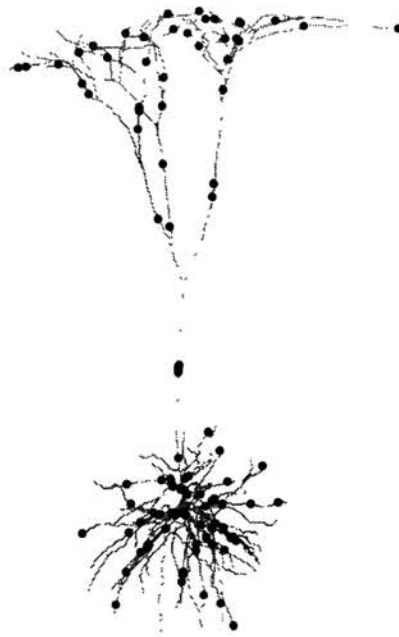

Figure 1: Layer 5 pyramidal neuron used in the simulations, showing 100 synaptic contacts. Morphology courtesy Rodney Douglas and Kevan Martin.

excitatory conductances (see Table 1). Conductances were scaled by an estimate of the local input resistance, to keep local EPSP size approximately uniform across the dendritic tree. Inhibitory synapses were not modeled.

## 2.2  MAPPING VISUAL STIMULI ONTO THE DENDRITIC TREE

A stimulus image consisted of a 64 × 64 pixel array containing a light or dark bar (pixel value ±1 against a background of 0). Bars of length 45 and width 7 were presented at various orientations and positions within the image. Images were linearly filtered through difference-of-Gaussian receptive fields (center width: 0.6, surround width: 1.2, with no DC response). Filtered images were then mapped onto 64 × 64 arrays of ON-center and OFF-center LGN cells, whose outputs were thresholded at ±0.02 respectively. In a crude model of gain control, only a random subset of 100 of the LGN neurons remained active to drive the modeled cortical cell.

Each LGN neuron gave rise to a single synapse onto the cortical cell's dendritic tree. In a given run, excitatory synapses originating from the 100 active LGN cells were activated asynchronously at 40 Hz, while all other synapses remained silent.

The spatial arrangement of connections from LGN cells onto the pyramidal cell dendrites was generated automatically, such that pairs of LGN cells which are co-active during presentations of optimally oriented bars formed synapses at nearby sites in the dendritic tree. The activity of the LGN cell array to an optimally oriented bar is shown in fig. 3. Frequently co-activated pairs of LGN neurons are hereafter referred to as "friend-pairs", and lie in a geometric arrangement as shown in fig. 4. Correlation-based clustering of friend-pairs was achieved by (1) choosing a random LGN cell and placing it at the next available dendritic site, (2) randomly

| Parameter | Value |
|---|---|
| $R_m$ | $10\text{k}\Omega\text{cm}^2$ |
| $R_a$ | $200\Omega\text{cm}$ |
| $C_m$ | $1.0\mu\text{F}/\text{cm}^2$ |
| $V_{\text{rest}}$ | -70 mV |
| Somatic $\bar{g}_{\text{Na}}$ | $0.20$ S/cm$^2$ |
| Somatic $\bar{g}_{\text{DR}}$ | $0.12$ S/cm$^2$ |
| Synapse count | 100 |
| Stimulus frequency | 40 Hz |
| $\tau_{\text{AMPA}}(on, off)$ | 0.5 ms, 3 ms |
| $\bar{g}_{\text{AMPA}}$ | 0.27 nS – 2.95 nS |
| $\tau_{\text{NMDA}}(on, off)$ | 0.5 ms, 50 ms |
| $\bar{g}_{\text{NMDA}}$ | 0.027 nS – 0.295 nS |
| $E_{\text{syn}}$ | 0 mV |

Figure 2: Table 1. Simulation Parameters.

choosing one of its friends and placing it at the next available dendritic site, and so on, until until either all of the cell's friends had already been deployed, in which case a new cell was chosen at random to restart the sequence, or all cells had been chosen, meaning that all of the 8192 (= 64 × 64 × 2) LGN synapses had been successfully mapped onto the dendritic tree. In previous modeling work it was shown that this type of clustering of correlated inputs on dendrites is the natural outcome of a balance between activity-independent synapse formation, and activity dependent synapse stabilization [6].

This method guaranteed that an optimally oriented bar stimulus activated a larger number of friend-pairs on average than did bars at non-optimal orientations. This led in turn to relatively clustery distributions of activated synapses in the dendrites in response to optimal bar orientations, in comparison to non-optimal orientations. In previous work, it was shown that synapses activated in clusters about a dendritic arbor could produce significantly larger cell responses than the same number of synapses activated diffusely about the dendritic tree [7, 8].

## 3   Results

Results for two series of runs are shown in fig. 5. For each bar stimulus, average spike rate was measured over a 250 ms period, beginning with the first spike initiated after stimulus onset (if any). This measure de-emphasized the initial transient climb off the resting potential, and provided a rough steady-state measure of stimulus effectiveness. Spike rates for 30 runs were averaged for each input condition.

Orientation tuning curves for a thin bar (7 × 45 pixels) are shown in fig. 5. The orientation tuning peaks sharply within about 10° of vertical, and then decays slowly for larger angles. Tuning is apparent both for dark and light bars, and remains independent of location within the receptive field.

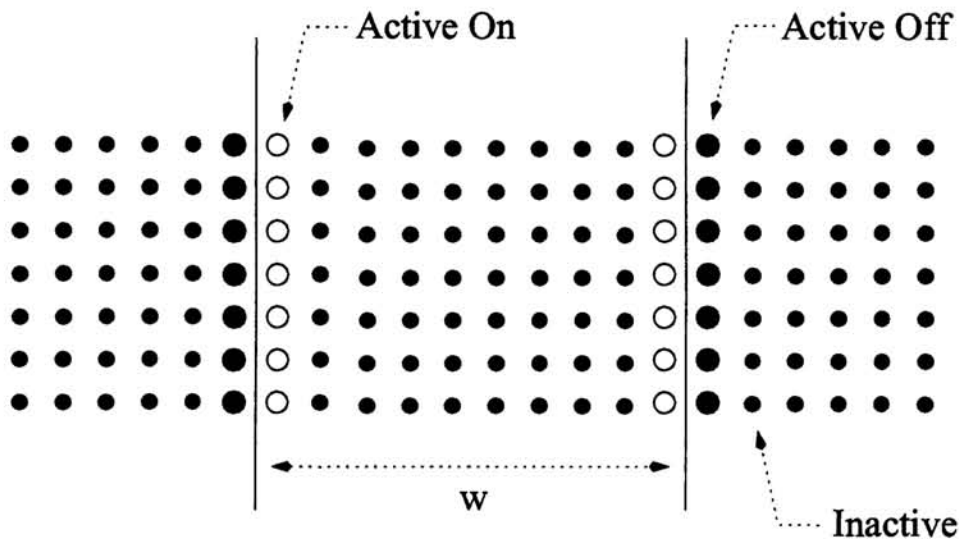

Figure 3: LGN cell activities in response to a vertical light bar of width $w = 10$ presented against a dark background. Large white circles: active on-center cells; large dark circles: active off-center cells; small gray circles: inactive cells. Only a $22 \times 7$ section of the array is shown.

## 4   Discussion

The results of fig. 5 indicate that a pyramidal cell driven exclusively by excitatory inputs from ON- and OFF-center LGN cells, is at a biophysical level *capable* of producing the hallmark nonlinear response property of visual complex cells. Furthermore, the cell's translation-invariant preference for light or dark vertical bars was established by manipulating only the spatial arrangement of connections from LGN cells onto the pyramidal cell dendrites. Since exactly 100 synapses were activated in every tested condition, the significantly larger responses to optimal bar orientations could not be explained by a simple elevation in the total synaptic activity impinging on the neuron in that condition. The origin of the cell's orientation-selective response resulted from nonlinear pooling of a large number of minimally-oriented subunits, i.e. consisting of pairs of ON and OFF cells that were co-consistent with an optimally oriented bar. We have achieved similar results in other experiments with a variety of different friend-neighborhood structures including ones both simpler and more complex than were used here, for LGN arrays with substantially different degrees of receptive field overlap, with random subsampling of the LGN array, with graded LGN activity levels, and for dendritic trees containing active sodium channels in addition to NMDA channels.

Thus far we have not attempted to relate physiologically-measured orientation and width tuning curves, and other detailed aspects of complex cell physiology, to our model cell, as we have been principally interested in establishing whether the most salient nonlinear features of complex cell physiology were biophysically feasible at the single cell level. Detailed comparisons between our results and empirical tuning curves, etc., must be made with caution, since our model cell has been "explanted" from the network in which it normally exists, and is therefore absent the normal recurrent excitatory and inhibitory influences the cortical network provides.

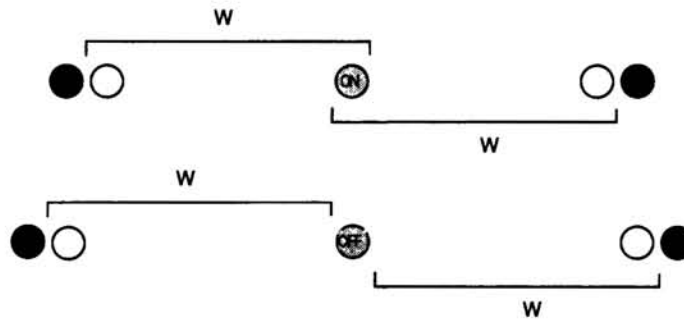

Figure 4: Layout of friends for an ON-center LGN cell for vertically oriented thin bars (top). The linear friendship linkage for a given ideal vertical bar of width $w$ was determined as follows. Suppose an LGN cell is chosen at random, e.g. an ON-center cell at location $(i, j)$ within the cell array. When a vertical bar is presented, LGN cells along the two vertical edges of the bar become active. The ON-center cell at position $(i, j)$ is active to a light bar when it is in a column of cells just inside either edge of the bar. Those cells which are co-active under this circumstance are: (a) other on-center cells in the same vertical column, (b) on-center cells in vertical columns a distance $w - 1$ to the right and left (depending on the bar position), (c) off-center cells in columns a distance $\pm 1$ away (due to the negative-going edge adjacent), and (d) off-center cells a distance $w$ to the right and left (due to the opposite edge). As "friend-pairs" we take only those LGN cells a distance $\pm(w - 1)$ and $\pm w$ away. Those in the same and neighboring columns are not included. The friends of an off-center cell are shown in the bottom figure. It and its friends are optimally stimulated by bars of width $w$ placed as shown. The width selected for our friend-pairs was $w = 7$, the same width as all bars presented as stimuli.

Experimental validation of these simulation results would imply a significant change in our conception of the role of the single neuron in neocortical processing.

## Acknowledgments

This work was funded by grants from the National Science Foundation and the Office of Naval Research.

## References

[1] G.M. Ghose, R.D. Freeman, and I. Ohzawa. Local intracortical connections in the cats visual-cortex - postnatal-development and plasticity. *J. Neurophysiol.*, 72:1290–1303, 1994.

[2] D.J. Heeger. Normalization of cell responses in cat striate cortex. *Visual Neurosci.*, 9:181–197, 1992.

[3] K.P. Hoffman and J. Stone. Conduction velocity of afferents to cat visual cortex: a correlation with cortical receptive field properties. *Brain Res.*, 32:460–466, 1971.

[4] D.H. Hubel and T.N. Wiesel. Receptive fields, binocular interaction and functional architecture in the cat's visual cortex. *J. Physiol.*, 160:106–154, 1962.

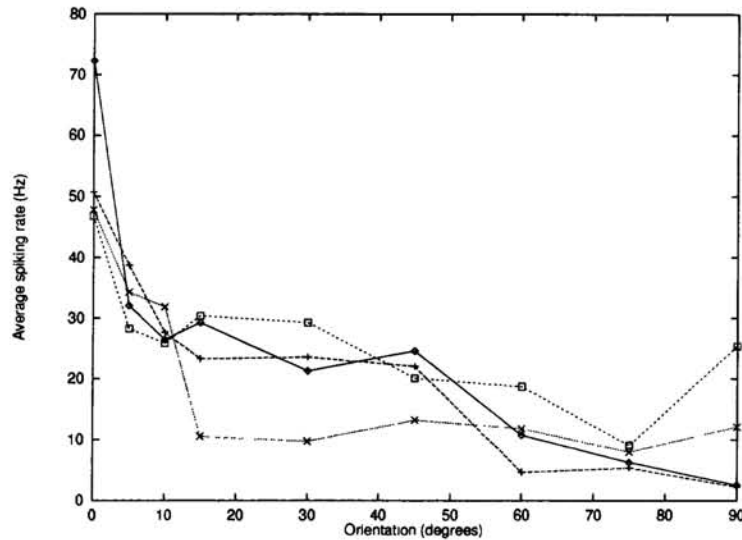

Figure 5: Orientation tuning curves for the model neurons. 'X': light bars centered in the receptive field; diamonds: light bars displaced by 6 pixels horizontally; squares: dark bars centered in the receptive field; '+': dark bars displaced by 6 pixels. Standard errors on the data are about 5 spikes/sec.

[5] J.G. Malpeli, C. Lee, H.D. Schwark, and T.G. Weyand. Cat area 17. I. Pattern of thalamic control of cortical layers. *J. Neurophyiol.*, 46:1102–1119, 1981.

[6] B.W. Mel. The clusteron: Toward a simple abstraction for a complex neuron. In J. Moody, S. Hanson, and R. Lippmann, editors, *Advances in Neural Information Processing Systems, vol. 4*, pages 35–42. Morgan Kaufmann, San Mateo, CA, 1992.

[7] B.W. Mel. NMDA-based pattern discrimination in a modeled cortical neuron. *Neural Computation*, 4:502–516, 1992.

[8] B.W. Mel. Synaptic integration in an excitable dendritic tree. *J. Neurophysiol.*, 70(3):1086–1101, 1993.

[9] B.W. Mel. Information processing in dendritic trees. *Neural Computation*, 6:1031–1085, 1994.

[10] J.A. Movshon. The velocity tuning of single units in cat striate cortex. *J. Physiol. (Lond)*, 249:445–468, 1975.

[11] I. Ohzawa, G.C. DeAngelis, and R.D Freeman. Stereoscopic depth discrimination in the visual cortex: Neurons ideally suited as disparity detectors. *Science*, 279:1037–1041, 1990.

[12] D. Pollen and S. Ronner. Visual cortical neurons as localized spatial frequency filters. *IEEE Trans. Sys. Man Cybern.*, 13:907–916, 1983.

[13] H.R. Wilson, D. Levi, L. Maffei, J. Rovamo, and R. DeValois. The perception of form: retina to striate cortex. In L. Spillman and J.S. Werner, editors, *Visual perception: the neurophysiological foundations*, pages 231–272. Academic Press, San Diego, 1990.